# On the Efficient Minimization of Classification Calibrated Surrogates

**Richard Nock**
CEREGMIA — Univ. Antilles-Guyane
97275 Schoelcher Cedex, Martinique, France
rnock@martinique.univ-ag.fr

**Frank Nielsen**
LIX - Ecole Polytechnique
91128 Palaiseau Cedex, France
nielsen@lix.polytechnique.fr

## Abstract

Bartlett *et al* (2006) recently proved that a ground condition for convex surrogates, classification calibration, ties up the minimization of the surrogates and classification risks, and left as an important problem the algorithmic questions about the minimization of these surrogates. In this paper, we propose an algorithm which provably minimizes any classification calibrated surrogate strictly convex and differentiable — a set whose losses span the exponential, logistic and squared losses —, with boosting-type guaranteed convergence rates under a weak learning assumption. A particular subclass of these surrogates, that we call balanced convex surrogates, has a key rationale that ties it to maximum likelihood estimation, zero-sum games and the set of losses that satisfy some of the most common requirements for losses in supervised learning. We report experiments on more than 50 readily available domains of 11 flavors of the algorithm, that shed light on new surrogates, and the potential of data dependent strategies to *tune* surrogates.

## 1 Introduction

A very active supervised learning trend has been flourishing over the last decade: it studies functions known as *surrogates* — upperbounds of the empirical risk, generally with particular convexity properties —, whose minimization remarkably impacts on empirical / true risks minimization [3, 4, 10]. Surrogates play fundamental roles in some of the most successful supervised learning algorithms, including AdaBoost, additive logistic regression, decision tree induction, Support Vector Machines [13, 7, 10]. As their popularity has been rapidly spreading, authors have begun to stress the need to set in order the huge set of surrogates, and better understand their properties. Statistical consistency properties have been shown for a wide set containing most of the surrogates relevant to learning, *classification calibrated surrogates* (CCS) [3]; other important properties, like the algorithmic questions about minimization, have been explicitly left as important problems to settle [3].

In this paper, we address and solve this problem for all strictly convex differentiable CCS, a set referred to as *strictly convex surrogates* (SCS). We propose a minimization algorithm, ULS, which outputs linear separators, with two key properties: it provably achieves the optimum of the surrogate, and meets Boosting-type convergence under a weak learning assumption. There is more, as we show that SCS strictly contains another set of surrogates of important rationale, *balanced convex surrogates* (BCS). This set, which contains the logistic and squared losses but not the exponential loss, coincides with the set of losses satisfying three common requirements about losses in learning. In fact, BCS spans a large subset of the expected losses for zero-sum games of [9], by which ULS may also be viewed as an efficient learner for decision making (in simple environments, though).

Section 2 gives preliminary definitions; section 3 presents surrogates losses and risks; sections 4 and 5 present ULS and its properties; section 6 discusses experiments with ULS; section 7 concludes.

## 2  Preliminary definitions

Unless otherwise stated, bold-faced variables like $\boldsymbol{w}$ denote vectors (components are $w_i, i = 1, 2, ...$), calligraphic upper-cases like $\mathcal{S}$ denote sets, and blackboard faces like $\mathbb{O}$ denote subsets of $\mathbb{R}$, the set of real numbers. We let set $\mathcal{O}$ denote a *domain* ($\mathbb{R}^n$, $[0,1]^n$, etc., where $n$ is the number of description variables), whose elements are *observations*. An *example* is an ordered pair $(\boldsymbol{o}, c) \in \mathcal{O} \times \{c^-, c^+\}$, where $\{c^-, c^+\}$ denotes the set of classes (or *labels*), and $c^+$ (resp. $c^-$) is the *positive* class (resp. *negative* class). Classes are abstracted by a bijective mapping to one of two other sets:

$$c \in \{c^-, c^+\} \leftrightharpoons y^* \in \{-1, +1\} \leftrightharpoons y \in \{0, 1\} \ . \tag{1}$$

The convention is $c^+ \rightleftharpoons +1 \rightleftharpoons 1$ and $c^- \rightleftharpoons -1 \rightleftharpoons 0$. We thus have three distinct notations for an example: $(\boldsymbol{o}, c)$, $(\boldsymbol{o}, y^*)$, $(\boldsymbol{o}, y)$, that shall be used without ambiguity. We suppose given a set of $m$ examples, $\mathcal{S} = \{(\boldsymbol{o}_i, c_i), i = 1, 2, ..., m\}$. We wish to build a *classifier* $H$, which can either be a function $H : \mathcal{O} \rightarrow \mathbb{O} \subseteq \mathbb{R}$ (hereafter, $\mathbb{O}$ is assumed to be symmetric with respect to 0), or a function $H : \mathcal{O} \rightarrow [0, 1]$. Following a convention of [6], we compute to which extent the outputs of $H$ and the labels in $\mathcal{S}$ disagree, $\varepsilon(\mathcal{S}, H)$, by summing a *loss* which quantifies pointwise disagreements:

$$\varepsilon(\mathcal{S}, H) \ \doteq \ \sum_i \ell(c_i, H(\boldsymbol{o}_i)) \ . \tag{2}$$

The fundamental loss is the *0/1 loss*, $\ell^{0/1}(c, H)$ (to ease readability, the second argument is written $H$ instead of $H(\boldsymbol{o})$). It takes on two forms depending on $\mathrm{im}(H)$:

$$\ell_{\mathbb{R}}^{0/1}(y^*, H) \doteq 1_{y^* \neq \sigma \circ H} \text{ if } \mathrm{im}(H) = \mathbb{O} \ , \text{ or } \ell_{[0,1]}^{0/1}(y, H) \doteq 1_{y \neq \tau \circ H} \text{ if } \mathrm{im}(H) = [0,1] \ . \tag{3}$$

The following notations are introduced in (3): for a clear distinction of the output of $H$, we put in index to $\ell$ and $\varepsilon$ an indication of the loss' domain of parameters: $\mathbb{R}$, meaning it is actually some $\mathbb{O} \subseteq \mathbb{R}$, or $[0,1]$. The exponent to $\ell$ gives the indication of the loss name. Finally, $1_\pi$ is the indicator variable that takes value 1 iff predicate $\pi$ is **true**, and 0 otherwise; $\sigma : \mathbb{R} \rightarrow \{-1, +1\}$ is $+1$ iff $x \geq 0$ and $-1$ otherwise; $\tau : [0, 1] \rightarrow \{0, 1\}$ is 1 iff $x \geq 1/2$, and 0 otherwise.

Both losses $\ell_{\mathbb{R}}$ and $\ell_{[0,1]}$ are defined simultaneously via popular transforms on $H$, such as the *logit* transform $\mathrm{logit}(p) \doteq \log(p/(1 - p)), \forall p \in [0, 1]$ [7]. We have indeed $\ell_{[0,1]}^{0/1}(y, H) = \ell_{\mathbb{R}}^{0/1}(y^*, \mathrm{logit}(H))$ and $\ell_{\mathbb{R}}^{0/1}(y^*, H) = \ell_{[0,1]}^{0/1}(y, \mathrm{logit}^{-1}(H))$. We have implicitly closed the domain of the logit, adding two symbols $\pm\infty$ to ensure that the eventual infinite values for $H$ can be mapped back to $[0, 1]$. In supervised learning, the objective is to carry out the minimization of the expectation of the 0/1 loss in *generalization*, the so-called *true risk*. Very often however, this task can be relaxed to the minimization of the *empirical risk* of $H$, which is (2) with the 0/1 loss [6]:

$$\varepsilon^{0/1}(\mathcal{S}, H) \ \doteq \ \sum_i \ell^{0/1}(c_i, H(\boldsymbol{o}_i)) \ . \tag{4}$$

The main classifiers we investigate are linear separators (LS). In this case, $H(\boldsymbol{o}) \doteq \sum_t \alpha_t h_t(\boldsymbol{o})$ for features $h_t$ with $\mathrm{im}(h_t) \subseteq \mathbb{R}$ and leveraging coefficients $\alpha_t \in \mathbb{R}$.

## 3  Losses and surrogates

A serious alternative to directly minimizing (4) is to rather focus on the minimization of a *surrogate risk* [3]. This is a function $\varepsilon(\mathcal{S}, H)$ as in (2) whose *surrogate loss* $\ell(c, H(\boldsymbol{o}))$ satisfies $\ell^{0/1}(c, H(\boldsymbol{o})) \leq \ell(c, H(\boldsymbol{o}))$. Four are particularly important in supervised learning, defined via the following surrogate losses:

$$\ell_{\mathbb{R}}^{\exp}(y^*, H) \ \doteq \ \exp(-y^* H) \ , \tag{5}$$

$$\ell_{\mathbb{R}}^{\log}(y^*, H) \ \doteq \ \log(1 + \exp(-y^* H)) \ , \tag{6}$$

$$\ell_{\mathbb{R}}^{\mathrm{sqr}}(y^*, H) \ \doteq \ (1 - y^* H)^2 \ , \tag{7}$$

$$\ell_{\mathbb{R}}^{\mathrm{hinge}}(y^*, H) \ \doteq \ \max\{0, 1 - y^* H\} \ . \tag{8}$$

(5) is the exponential loss, (6) is the logistic loss, (7) is the squared loss and (8) is hinge loss.

**Definition 1** *A Strictly Convex Loss (SCL) is a strictly convex function $\psi : \mathbb{X} \rightarrow \mathbb{R}_+$ differentiable on $\mathrm{int}(\mathbb{X})$ with $\mathbb{X}$ symmetric interval with respect to zero, s. t. $\nabla_\psi(0) < 0$.*

| $\phi(x)$ | $a_\phi$ | $\text{im}(\nabla_{\overline{\phi}})$ $\supseteq \text{im}(H)$ | $F_\phi(y^*H)$ $= (\overline{\phi}^\star(-y^*H) - a_\phi)/b_\phi$ | $\hat{\mathbf{Pr}}[c = c^+\|H; o]$ $= \nabla_{\overline{\phi}}^{-1}(H)$ |
|---|---|---|---|---|
| $\phi_{\mu,\mu\in(0,1)}(x) \doteq \mu + (1-\mu)\sqrt{x(1-x)}$ | $\mu$ | $\mathbb{R}$ | $\frac{-y^*H + \sqrt{(1-\mu)^2 + (y^*H)^2}}{1-\mu}$ | $\frac{1}{2} + \frac{H}{2\sqrt{(1-\mu)^2 + H^2}}$ |
| $\phi_{\mathrm{M}}(x) \doteq \sqrt{x(1-x)}$ | $0$ | $\mathbb{R}$ | $-y^*H + \sqrt{1 + (y^*H)^2}$ | $\frac{1}{2} + \frac{H}{2\sqrt{1+H^2}}$ |
| $\phi_{\mathrm{Q}}(x) \doteq -x\log x - (1-x)\log(1-x)$ | $0$ | $\mathbb{R}$ | $\log(1 + \exp(-y^*H))$ | $\frac{\exp(H)}{1+\exp(H)}$ |
| $\phi_{\mathrm{B}}(x) \doteq x(1-x)$ | $0$ | $[-1,1]$ | $(1-y^*H)^2$ | $\frac{1}{2} + \frac{H}{2}$ |

Table 1: permissible functions, their corresponding BCLs and the matching $[0,1]$ predictions.

$\nabla_.$ is the gradient notation (here, the derivative). Any surrogate risk built from a SCL is called a Strictly Convex Surrogate (SCS). From Theorem 4 in [3], it comes that SCL contains all classification calibrated losses (CCL) that are strictly convex and differentiable, such as (5), (6), (7).

Fix $\psi \in$ SCL. The *Legendre conjugate* $\psi^\star$ of $\psi$ is $\psi^\star(x) \doteq \sup_{x' \in \text{int}(\mathbb{X})}\{xx' - \psi(x')\}$. Because of the strict convexity of $\psi$, the analytic expression of the Legendre conjugate becomes $\psi^\star(x) \doteq x\nabla_\psi^{-1}(x) - \psi(\nabla_\psi^{-1}(x))$. $\psi^\star$ is also strictly convex and differentiable. A function $\phi : [0,1] \to \mathbb{R}_+$ is called permissible iff it is differentiable on $(0,1)$, strictly concave, symmetric about $x = 1/2$, and with $\phi(0) = \phi(1) = a_\phi \geq 0$. We let $b_\phi \doteq \phi(1/2) - a_\phi > 0$. Permissible functions with $a_\phi = 0$ span a very large subset of generalized entropies [9]. Permissible functions are useful to define the following subclass of SCL, of particular interest (here, $\overline{\phi} \doteq -\phi$).

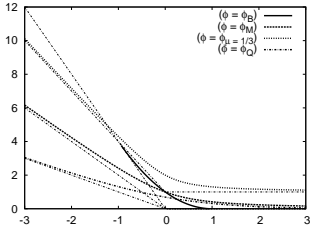

Figure 1: Bold curves depict plots of $\overline{\phi}^\star(-x)$ for the $\phi$ in Table 1; thin dotted half-lines are its asymptotes.

**Definition 2** *Let $\phi$ permissible. The Balanced Convex Loss (BCL) with signature $\phi$, $F_\phi$, is:*

$$F_\phi(x) \doteq (\overline{\phi}^\star(-x) - a_\phi)/b_\phi . \qquad (9)$$

Balanced Convex Surrogates (BCS) are defined accordingly. All BCL share a common shape. Indeed, $\overline{\phi}^\star(x)$ satisfies the following relationships:

$$\overline{\phi}^\star(x) = \overline{\phi}^\star(-x) + x , \qquad (10)$$
$$\lim_{x \to \text{infim}(\nabla_{\overline{\phi}})} \overline{\phi}^\star(x) = a_\phi . \qquad (11)$$

Noting that $F_\phi(0) = 1$ and $\nabla_{F_\phi}(0) = -(1/b_\phi)\nabla_{\overline{\phi}}^{-1}(0) < 0$, it follows that BCS $\subset$ SCS, where the strict inequality comes from the fact that (5) is a SCL but not a BCL. It also follows $\lim_{x \to \text{supim}(\nabla_{\overline{\phi}})} F_\phi(x) = 0$ from (11), and $\lim_{x \to \text{infim}(\nabla_{\overline{\phi}})} F_\phi(x) = -x/b_\phi$ from (10). We get that the asymptotes of any BCL can be summarized as $\underline{\ell}(x) \doteq x(\sigma(x) - 1)/(2b_\phi)$. When $b_\phi = 1$, this is the linear hinge loss [8], a generalization of (8) for which $x \doteq y^*H - 1$. Thus, while hinge loss is not a BCL, it is the limit behavior of any BCL (see Figure 1).

Table 1 (left column) gives some examples of permissible $\phi$. When scaled so that $\phi(1/2) = 1$, some confound with popular choices: $\phi_{\mathrm{B}}$ with Gini index, $\phi_{\mathrm{Q}}$ with the Bit-entropy, and $\phi_{\mathrm{M}}$ with Matsushita's error [10, 11]. Table 1 also gives the expressions of $F_\phi$ along with the $\text{im}(H) = \mathbb{O} \subseteq \mathbb{R}$ allowed by the BCL, for the corresponding permissible function. It is interesting to note the constraint on $\text{im}(H)$ for the squared loss to be a BCL, which makes it monotonous in the interval, but implies to rescale the outputs of classifiers like linear separators to remain in $[-1,1]$.

## 4 ULS: the efficient minimization of any SCS

For any strictly convex function $\psi : \mathbb{X} \to \mathbb{R}$ differentiable on $\text{int}(\mathbb{X})$, the *Bregman Loss Function* (BLF) $D_\psi$ with generator $\psi$ is [5]:

$$D_\psi(x\|x') \doteq \psi(x) - \psi(x') - (x - x')\nabla_\psi(x') . \qquad (12)$$

The following Lemma states some relationships that are easy to check using $\psi^{\star\star} = \psi$. They are particularly interesting when $\text{im}(H) = \mathbb{O} \subseteq \mathbb{R}$.

---

**Algorithm 1:** Algorithm ULS$(M, \psi)$

---

**Input**: $M \in \mathbb{R}^{m \times T}$, SCL $\psi$ with $\mathrm{dom}(\psi) = \mathbb{R}$;

Let $\boldsymbol{\alpha}_1 \leftarrow \mathbf{0}$; Let $\boldsymbol{w}_0 \leftarrow \nabla_{\tilde{\psi}}^{-1}(0)\mathbf{1}$;

**for** $j = 1, 2, ...J$ **do**
   |    **[WU]** (weight update) $\boldsymbol{w}_j \leftarrow (M\boldsymbol{\alpha}_j) \diamond \boldsymbol{w}_0$ ;
   |    Let $\mathcal{T}_j \subseteq \{1, 2, ..., T\}$; let $\boldsymbol{\delta}_j \leftarrow \mathbf{0}$;
   |    **[LC]** (leveraging coefficients) $\forall t \in \mathcal{T}_j$, pick $\delta_{j,t}$ such that: $\sum_{i=1}^{m} m_{it}((M\boldsymbol{\delta}_j) \diamond \boldsymbol{w}_j)_i = 0$ ;
   |    Let $\boldsymbol{\alpha}_{j+1} \leftarrow \boldsymbol{\alpha}_j + \boldsymbol{\delta}_j$;

**Output**: $H(\boldsymbol{x}) \doteq \sum_{t=1}^{T} \alpha_{J+1,t} h_t(\boldsymbol{x}) \in \mathrm{LS}$

---

**Lemma 1** *For any* SCL $\psi$, $\psi(y^*H) = D_{\psi^\star}(0||\nabla_{\psi^\star}^{-1}(y^*H)) - \psi^\star(0)$. *Furthermore, for any* BCL $F_\phi$, $D_{\overline{\phi}}(y||\nabla_{\overline{\phi}}^{-1}(H)) = b_\phi F_\phi(y^*H)$ *and* $D_{\overline{\phi}}(y||\nabla_{\overline{\phi}}^{-1}(H)) = D_{\overline{\phi}}(1||\nabla_{\overline{\phi}}^{-1}(y^*H))$.

The second equality is important because it ties real predictions (right) with $[0, 1]$ predictions (left). It also separates SCL and BCL, as for any $\psi$ in SCL, it can be shown that there exists a functions $\varphi$ such that $D_\varphi(y||\nabla_\varphi^{-1}(H)) = \psi(y^*H)$ iff $\psi \in$ BCL. We now focus on the minimization of any SCS. We show that there exists an algorithm, ULS, which fits a linear separator $H$ to the minimization of any SCS $\varepsilon_{\mathbb{R}}^\psi \doteq \sum_i \psi(y_i^* H(\boldsymbol{o}_i))$ for any SCL $\psi$ with $\mathrm{dom}(\psi) = \mathbb{R}$, in order not to restrict the LS built. To simplify notations, we let:

$$\tilde{\psi}(x) \quad \doteq \quad \psi^\star(-x) \ . \tag{13}$$

With this notation, the first equality in Lemma 1 becomes:

$$\psi(y^*H) \quad = \quad D_{\tilde{\psi}}(0||\nabla_{\tilde{\psi}}^{-1}(-y^*H)) - \tilde{\psi}(0) \ . \tag{14}$$

We let $\mathbb{W} \doteq \mathrm{dom}(\nabla_{\tilde{\psi}}) = -\mathrm{im}(\nabla_\psi)$, where this latter equality comes from $\nabla_{\tilde{\psi}}(x) = -\nabla_{\psi^\star}(-x) = -\nabla_\psi^{-1}(-x)$. It also comes $\mathrm{im}(\nabla_{\tilde{\psi}}) = \mathbb{R}$. Because any BLF is strictly convex in its first argument, we can compute its Legendre conjugate. In fact, we shall essentially need the argument that realizes the supremum: for any $x \in \mathbb{R}$, for any $p \in \mathbb{W}$, we let:

$$x \diamond p \quad \doteq \quad \arg_{p' \in \mathbb{W}} \sup\{xp' - D_{\tilde{\psi}}(p'||p)\} \ . \tag{15}$$

We do not make reference to $\tilde{\psi}$ in the $\diamond$ notation, as it shall be clear from context. We name $x \diamond p$ the Legendre *dual* of the ordered pair $(x, p)$, closely following a notation by [6]. The Legendre dual is unique and satisfies:

$$\nabla_{\tilde{\psi}}(x \diamond p) = x + \nabla_{\tilde{\psi}}(p) \ , \tag{16}$$

$$\forall x, x' \in \mathbb{R}, \forall p \in \mathbb{W}, x \diamond (x' \diamond p) = (x + x') \diamond p \ . \tag{17}$$

To state ULS, we follow the setting of [6] and suppose that we have $T$ features $h_t$ ($t = 1, 2, ..., T$) known in advance, the problem thus reducing to the computation of the leveraging coefficients. We define $m \times T$ matrix $M$ with:

$$m_{it} \quad \doteq \quad -y_i^* h_t(\boldsymbol{o}_i) \ . \tag{18}$$

Given leveraging coefficients vector $\boldsymbol{\alpha} \in \mathbb{R}^T$, we get:

$$-y_i^* H(\boldsymbol{o}_i) \quad = \quad (M\boldsymbol{\alpha})_i \ . \tag{19}$$

We can specialize this setting to classical greedy induction frameworks for LS: in classical boosting, at step $j$, we would fit a single $\alpha_t$ [6]; in totally corrective boosting, we would rather fit $\{\alpha_t, 1 \leq t \leq j\}$ [14]. Intermediate schemes may be used as well for $\mathcal{T}_j$, provided they ensure that, at each step $j$ of the algorithm and for any feature $h_t$, it may be chosen at some $j' > j$. ULS is displayed in Algorithm 1. In Algorithm 1, notations are vector-based: the Legendre duals are computed component-wise; furthermore, $\mathcal{T}_j$ may be chosen according to whichever scheme underlined above. The following Theorem provides a first general convergence property for ULS.

**Theorem 1** *ULS($M$, $\psi$) converges to a classifier $H$ realizing the minimum of $\varepsilon_{\mathbb{R}}^\psi$.*

**Proof sketch:** In step [WU] in ULS, (17) brings $\boldsymbol{w}_{j+1} = (M\boldsymbol{\alpha}_{j+1}) \diamond \boldsymbol{w}_0 = (M\boldsymbol{\delta}_j) \diamond \boldsymbol{w}_j$. After few derivations involving the choice of $\boldsymbol{\delta}_j$ and step [LC] in ULS, we obtain (with vector notations, BLFs are the sum of the component-wise BLFs):

$$D_{\tilde{\psi}}(\mathbf{0}||\boldsymbol{w}_{j+1}) - D_{\tilde{\psi}}(\mathbf{0}||\boldsymbol{w}_j) \quad = \quad -D_{\tilde{\psi}}(\boldsymbol{w}_{j+1}||\boldsymbol{w}_j) \tag{20}$$

Let $A_{\tilde{\psi}}(\boldsymbol{w}_{j+1}, \boldsymbol{w}_j) \doteq -D_{\tilde{\psi}}(\boldsymbol{w}_{j+1}||\boldsymbol{w}_j)$, which is just, from (20) and (14), the difference between two successive SCL in Algorithm 1. Thus, $A_{\tilde{\psi}}(\boldsymbol{w}_{j+1}, \boldsymbol{w}_j) < 0$ whenever $\boldsymbol{w}_{j+1} \neq \boldsymbol{w}_j$. Should we be able to prove that when ULS has converged, $\boldsymbol{w}_. \in \mathrm{Ker}M^\top$, this would make $A_{\tilde{\psi}}(\boldsymbol{w}_{j+1}, \boldsymbol{w}_j)$ an *auxiliary function* for ULS, which is enough to prove the convergence of ULS towards the optimum [6]. Thus, suppose that $\boldsymbol{w}_{j+1} = \boldsymbol{w}_j$ (ULS has converged). Suppose that $\mathcal{T}_j$ is a singleton (*e.g.* classical boosting scheme). In this case, $\boldsymbol{\delta}_j = \mathbf{0}$ and so $\forall t = 1, 2, ..., T, \sum_{i=1}^m m_{it}(\mathbf{0} \diamond \boldsymbol{w}_j)_i = \sum_{i=1}^m m_{it} \boldsymbol{w}_{j,i} = 0$, *i.e.* $\boldsymbol{w}_j^\top M = \boldsymbol{w}_{j+1}^\top M = \mathbf{0}^\top$, and $\boldsymbol{w}_j, \boldsymbol{w}_{j+1} \in \mathrm{Ker}M^\top$. The case of totally corrective boosting is simpler, as after the last iteration we would have $\boldsymbol{w}_{J+1} \in \mathrm{Ker}M^\top$. Intermediate choices for $\mathcal{T}_j \subset \{1, 2, ..., T\}$ are handled in the same way. $\square$

We emphasize the fact that Theorem 1 proves the convergence towards the global optimum of $\varepsilon_\mathbb{R}^\psi$, regardless of $\psi$. The optimum is defined by the LS with features in $M$ that realizes the smallest $\varepsilon_\mathbb{R}^\psi$. Notice that in practice, it may be a tedious task to satisfy exactly (20), in particular for totally corrective boosting [14].

ULS has the flavor of boosting algorithms, repeatedly modifying a set of weights $\boldsymbol{w}$ over the examples. In fact, this similarity is more than syntactical, as ULS satisfies two first popular algorithmic boosting properties, the first of which being that step [LC] in ULS is equivalent to saying that this LS has zero *edge* on $\boldsymbol{w}_{j+1}$ [14]. The following Lemma shows that this edge conditions is sound.

**Lemma 2** *Suppose that there does not exist some $h_t$ with all $m_{it}$ of the same sign, $\forall i = 1, 2, ..., m$. Then, for any choice of $\mathcal{T}_j$, step [LC] in ULS has always a finite solution.*

**Proof:** Let:

$$Z \quad \doteq \quad D_{\tilde{\psi}}(\mathbf{0}||(M\boldsymbol{\alpha}_{j+1}) \diamond \boldsymbol{w}_0) \ . \tag{21}$$

We have $Z = m\tilde{\psi}(0) + \sum_{i=1}^m \tilde{\psi}(-(M(\boldsymbol{\delta_j} + \boldsymbol{\alpha}_j))_i)$ from (14), a function convex in all leveraging coefficients. Define $|\mathcal{T}_j| \times |\mathcal{T}_j|$ matrix $E$ with $e_{uv} \doteq \partial^2 Z/(\partial \delta_{j,u} \delta_{j,v})$ (for the sake of simplicity, $\mathcal{T}_j = \{1, 2, ..., |\mathcal{T}_j|\}$, where $|.|$ denotes the cardinal). We have $e_{uv} = \sum_{i=1}^m m_{iu} m_{iv}/\varphi(((M\boldsymbol{\delta}_j) \diamond \boldsymbol{w}_j)_i)$, with $\varphi(x) \doteq \mathrm{d}^2\tilde{\psi}(x)/\mathrm{d}x^2$ a function strictly positive in $\mathrm{int}(\mathbb{W})$ since $\tilde{\psi}$ is strictly convex. Let $q_{i,j} \doteq 1/\varphi(((M\boldsymbol{\delta}_j) \diamond \boldsymbol{w}_j)_i) > 0$. It is easy to show that $\boldsymbol{x}^\top E \boldsymbol{x} = \sum_{i=1}^m q_{i,j} \langle \boldsymbol{x}, \tilde{\boldsymbol{m}}_i \rangle^2 \geq 0, \forall \boldsymbol{x} \in \mathbb{R}^{|\mathcal{T}_j|}$, with $\tilde{\boldsymbol{m}}_i \in \mathbb{R}^{|\mathcal{T}_j|}$ the vector with $\tilde{m}_{it} \doteq m_{it}$. Thus, $E$ is positive semidefinite; as such, step [LC] in ULS, which is the same as solving $\partial Z/\partial \delta_{j,u} = 0, \forall u \in \mathcal{T}_j$ (*i.e.* minimizing $Z$) has always a solution. $\square$

The condition for the Lemma to work is absolutely not restrictive, as if such an $h_t$ were to exist, we would not need to run ULS: indeed, we would have either $\varepsilon^{0/1}(\mathcal{S}, h_t) = 0$, or $\varepsilon^{0/1}(\mathcal{S}, -h_t) = 0$. The second property met by ULS is illustrated in the second example below.

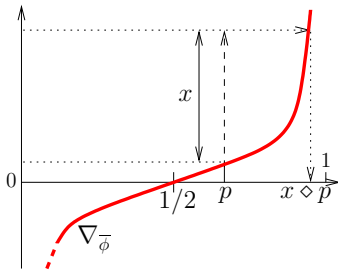

Figure 2: A typical $\nabla_{\overline{\phi}}$ (red: strictly increasing, symmetric wrt point $(1/2, 0)$), with Legendre dual $x \diamond p$ computed from $x$ and $p$.

We give two examples of specializations of ULS. Take for example $\psi(x) = \exp(-x)$ (5). In this case, $\mathbb{W} = \mathbb{R}_+$, $\boldsymbol{w}_0 = \mathbf{1}$ and it is not hard to see that ULS matches real AdaBoost with unnormalized weights [13]. The difference is syntactical: the LS output by ULS and real AdaBoost are the same. Now, take any BCL. In this case, $\tilde{\psi} = \overline{\phi}$, $\mathbb{W} = [0, 1]$ (scaling issues underlined for the logit in Section 2 make it desirable to close $\mathbb{W}$), and $\boldsymbol{w}_0 = 1/2 \mathbf{1}$. In all these cases, where $\mathbb{W} \subseteq \mathbb{R}_+$, $\boldsymbol{w}_j$ is always a distribution up to a normalization factor, and this would also be the case for any strictly monotonous SCS $\psi$. The BCL case brings an appealing display of how the weights behave. Figure 2 displays a typical Legendre dual for a BCL. Consider example $(\boldsymbol{o}_i, y_i)$, and its weight update, $w_{j,i} \leftarrow (M\boldsymbol{\alpha}_j)_i \diamond w_{0,i} = (-y_i^* H(\boldsymbol{o}_i)) \diamond w_{0,i}$ for the current classifier $H$. Fix $p = w_{0,i}$ and $x = -y_i^* H(\boldsymbol{o}_i)$ in Figure 2. We see that the new weight of the example gets larger iff $x > 0$, *i.e.* iff the example is given the *wrong* class by $H$, which is the second boosting property met by ULS.

ULS turns out to meet a third boosting property, and the most important as it contributes to root the algorithm in the seminal boosting theory of the early nineties: we have guarantees on its convergence rate under a generalization of the well-known "Weak Learning Assumption" (WLA) [13]. To state the WLA, we plug the iteration in the index of the distribution normalization coefficient in (21), and define $Z_j \doteq ||\boldsymbol{w}_j||_1$ ($||.||_k$ is the $L_k$ norm). The WLA is:

$$(\textbf{WLA})\forall j, \exists \gamma_j > 0 : |(1/|\mathcal{T}_j|) \sum_{t \in \mathcal{T}_j} (1/Z_j) \sum_{i=1}^m m_{it} w_{j,i}| \geq \gamma_j . \qquad (22)$$

This is indeed a generalization of the usual WLA for boosting algorithms, that we obtain taking $|\mathcal{T}_j| = 1$, $h_t \in \{-1, +1\}$ [12]. Few algorithms are known that formally boost WLA in the sense that requiring only WLA implies guaranteed rates for the minimization of $\varepsilon_{\mathbb{R}}^\psi$. We show that ULS meets this property $\forall \psi \in$ SCL. To state this, we need few more definitions. Let $\boldsymbol{m}_t$ denote the $t^{th}$ column vector of $M$, $a_{\boldsymbol{m}} \doteq \max_t ||\boldsymbol{m}_t||_2$ and $a_Z \doteq \min_j Z_j$. Let $a_\gamma$ denote the average of $\gamma_j$ ($\forall j$), and $a_\varphi \doteq \min_{x \in \text{int}(\mathbb{W})} \varphi(x)$ ($\varphi$ defined in the proof of Lemma 2).

**Theorem 2** *Under the WLA, ULS terminates in at most $J = \mathcal{O}(ma_{\boldsymbol{m}}^2/(a_\varphi a_Z^2 a_\gamma^2))$ iterations.*

**Proof sketch:** We use Taylor expansions with Lagrange remainder for $\tilde{\psi}$, and then the mean-value theorem, and obtain that $\forall w, w + \Delta \in \mathbb{W}, \exists w^\star \in [\min\{w + \Delta, w\}, \max\{w + \Delta, w\}]$ such that $D_{\tilde{\psi}}(w + \Delta||w) = \Delta^2 \varphi(w^\star)/2 \geq (\Delta^2/2) a_\varphi \geq 0$. We use $m$ times this inequality with $w = w_{j,i}$ and $\Delta = (w_{j+1,i} - w_{j,i})$, sum the inequalities, combine with Cauchy - Schwartz and Jensen's inequalities, and obtain:

$$D_{\tilde{\psi}}(\boldsymbol{w}_{j+1}||\boldsymbol{w}_j) \geq a_\varphi(a_Z \gamma_j/(2a_{\boldsymbol{m}}))^2 . \qquad (23)$$

Using (20), we obtain that $D_{\tilde{\psi}}(\boldsymbol{0}||\boldsymbol{w}_{J+1}) - m\tilde{\psi}(0)$ equals:

$$-m\tilde{\psi}(0) + D_{\tilde{\psi}}(\boldsymbol{0}||\boldsymbol{w}_1) + \sum_{j=1}^J (D_{\tilde{\psi}}(\boldsymbol{0}||\boldsymbol{w}_{j+1}) - D_{\tilde{\psi}}(\boldsymbol{0}||\boldsymbol{w}_j)) = m\psi(0) - \sum_{j=1}^J D_{\tilde{\psi}}(\boldsymbol{w}_{j+1}||\boldsymbol{w}_j) (24)$$

But, (14) together with the definition of $\boldsymbol{w}_j$ in [WU] (see ULS) yields $D_{\tilde{\psi}}(0||w_{J+1,i}) = \tilde{\psi}(0) + \psi(y_i^* H(\boldsymbol{o}_i)), \forall i = 1, 2, ..., m$, which ties up the SCS to (24); the guaranteed decrease in the rhs of (24) by (23) makes that there remains to check when the rhs becomes negative to conclude that ULS has terminated. This gives the bound of the Theorem. □
The bound in Theorem 2 is mainly useful to prove that the WLA guarantees a convergence rate of order $\mathcal{O}(m/a_\gamma^2)$ for ULS, but not the best possible as it is in some cases far from being optimal.

## 5 ULS, BCL, maximum likelihood and zero-sum games

BCL matches through the second equality in Lemma 1 the set of losses that satisfy the main requirements about losses used in machine learning. This is a strong rationale for its use. Suppose $\text{im}(H) \subseteq [0, 1]$, and consider the following requirements about some loss $\ell_{[0,1]}(y, H)$:

(**R1**) *The loss is lower-bounded.* $\exists z \in \mathbb{R}$ such that $\inf_{y,H} \ell_{[0,1]}(y, H) = z$.

(**R2**) *The loss is a proper scoring rule.* Consider a singleton domain $\mathcal{O} = \{\boldsymbol{o}\}$. Then, the best (constant) prediction is $\arg\min_{x \in [0,1]} \varepsilon_{[0,1]}(\mathcal{S}, x) = p \doteq \hat{\textbf{Pr}}[c = c^+|\boldsymbol{o}] \in [0, 1]$, where $p$ is the relative proportion of positive examples with observation $\boldsymbol{o}$.

(**R3**) *The loss is symmetric* in the following sense: $\ell_{[0,1]}(y, H) = \ell_{[0,1]}(1 - y, 1 - H)$.

**R1** is standard. For **R2**, we can write $\varepsilon_{[0,1]}(\mathcal{S}, x) = p\ell_{[0,1]}(1, x) + (1 - p)\ell_{[0,1]}(0, x) = L(p, x)$, which is just the expected loss of zero-sum games used in [9] (eq. (8)) with Nature states reduced to the class labels. The fact that the minimum is achieved at $x = p$ makes the loss a proper scoring rule. **R3** implies $\ell_{[0,1]}(1, 1) = \ell_{[0,1]}(0, 0)$, which is virtually assumed for any domain; otherwise, it scales to $H \in [0, 1]$ a well-known symmetry in the *cost matrix* that holds for domains without class dependent misclassification costs. For these domains indeed, it is assumed $\ell_{[0,1]}(1, 0) = \ell_{[0,1]}(0, 1)$.

Finally, we say that loss $\ell_{[0,1]}$ is *properly defined* iff $\mathrm{dom}(\ell_{[0,1]}) = [0,1]^2$ and it is twice differentiable on $(0,1)^2$. This is only a technical convenience: even the 0/1 loss coincides on $\{0,1\}$ with properly defined losses. In addition, the differentiability condition would be satisfied by many popular losses. The proof of the following Lemma involves Theorem 3 in [1] and additional facts to handle **R3**.

**Lemma 3** *Assume* $\mathrm{im}(H) \subseteq [0,1]$. *Loss* $\ell_{[0,1]}(.,.)$ *is properly defined and meets requirements **R1**, **R2**, **R3** iff* $\ell_{[0,1]}(y,H) = z + D_{\overline{\phi}}(y||H)$ *for some permissible* $\phi$.

Thus, $\phi$ maybe viewed as the "*signature*" of the loss. The second equality in Lemma 1 makes a tight connection between the predictions of $H$ in $[0,1]$ and $\mathbb{R}$. Let it be more formal: the *matching* $[0,1]$ prediction for some $H$ with $\mathrm{im}(H) = \mathbb{O}$ is:

$$\hat{\mathbf{Pr}}_\phi[c = c^+|H;\boldsymbol{o}] \quad \dot{=} \quad \nabla_{\overline{\phi}}^{-1}(H(\boldsymbol{o})) \ , \tag{25}$$

With this definition, illustrated in Table 1, Lemma 3 and the second equality in Lemma 1 show that BCL matches the set of losses of Lemma 3. This definition also brings the true nature of the minimization of any BCS with real valued hypotheses like linear separators (in ULS). From Lemma 3 and [2], there exists a bijection between BCL and a subclass of the exponential families whose members' pdfs may be written as: $\mathbf{Pr}_\phi[y|\theta] = \exp(-D_{\overline{\phi}}(y||\nabla_{\overline{\phi}}^{-1}(\theta)) + \overline{\phi}(y) - \nu(y))$, where $\theta \in \mathbb{R}$ is the natural parameter and $\nu(.)$ is used for normalization. Plugging $\theta = H(\boldsymbol{o})$, using (25) and the second equality in Lemma 1, we obtain that any BCS can be rewritten as $\varepsilon_{\mathbb{R}}^\phi = U + \sum_i - \log \mathbf{Pr}_\phi[y_i|H(\boldsymbol{o}_i)]$, where $U$ does not play a role in its minimization. We obtain the following Lemma, in which we suppose $\mathrm{im}(H) = \mathbb{O}$.

**Lemma 4** *Minimizing any* BCS *with classifier* $H$ *yields the maximum likelihood estimation, for each observation, of the natural parameter* $\theta = H(\boldsymbol{o})$ *of an exponential family defined by signature* $\phi$.

In fact, *one* exponential family is concerned *in fine*. To see this, we can factor the pdf as $\mathbf{Pr}[y|\theta] \dot{=} \exp(\theta\lambda(y) - \psi(\theta))/z$, with $\psi = \overline{\phi}^\star$ the cumulant function, $\lambda(y)$ the sufficient statistic and $z$ the normalization function. Since $y \in \{0,1\}$, we easily end up with $\mathbf{Pr}_\phi[y|\theta] = 1/(1 + \exp(-\theta))$, the logistic prediction for a Bernoulli prior. To summarize, minimizing any loss that meets **R1**, **R2** and **R3** (*i.e.* any BCL) amounts to the *same* ultimate goal; Since ULS works for any of the corresponding surrogate risks, the crux of the choice of the BCL relies on data-dependent considerations.

Finally, we can go further in the parallel with game theory developed above for **R2**: using notations in [9], the loss function of the decision maker can be written $L(X,q) = D_{\overline{\phi}}(1||q(X))$. **R3** makes it easy to recover losses like the log loss or the Brier score [9] respectively from $\phi_Q$ and $\phi_B$ (Table 1). In this sense, ULS is also a sound learner for decision making in the zero-sum game of [9]. Notice however that, to work, it requires that Nature has a restricted sample space size ($\{0,1\}$).

## 6   Experiments

We have compared against each other 11 flavors of ULS, including real AdaBoost [13], on a benchmark of 52 domains (49 from the UCI repository). True risks are estimated via stratified 10-fold cross validation; ULS is ran for $r$ (fixed) features $h_t$, each of which is a Boolean rule: **If** Monomial **then** Class$= \pm 1$ **else** Class $= \mp 1$, with at most $l$ (fixed) literals, induced following the greedy minimization of the BCS at hand. Leveraging coefficients ([LC] in ULS) are approximated up to $10^{-10}$ precision. Figure 3 summarizes the results for two values of the couple $(l,r)$. Histograms are ordered from left to right in increasing average true risk over all domains (shown below histograms). The *italic* numbers give, for each algorithm, the number of algorithms it *beats* according to a Student paired t-test over all domains with .1 threshold probability. Out of the 10 flavors of ULS, the first four flavors pick $\phi$ in Table 1. The fifth uses another permissible function: $\phi_\upsilon(x) \dot{=} (x(1-x))^\upsilon$ , $\forall \upsilon \in (0,1)$. The last five adaptively tune the BCS at hand out-of-a-bag of BCS. The first four fit the BCS at *each stage* of the inner loop (**for** $j$ ...) of ULS. Two (noted "$F.$") pick the BCS which minimizes the empirical risk in the bag; two others (noted "$E.$") pick the BCS which maximizes the current edge. There are two different bags corresponding to four permissible functions each: the first (index "1") contains the $\phi$ in Table 1, the second (index "2") replaces $\phi_B$ by $\phi_\upsilon$. We wanted to evaluate $\phi_B$ because it forces to renormalize the leveraging coefficients in $H$ each time it is selected, to ensure that the output of $H$ lies in $[-1,1]$. The last adaptive flavor, $F^*$, "externalizes" the choice of the BCS: it selects for each fold the BCS which yields the smallest empirical risk in a bag corresponding to five $\phi$: those of Table 1 plus $\phi_\upsilon$.

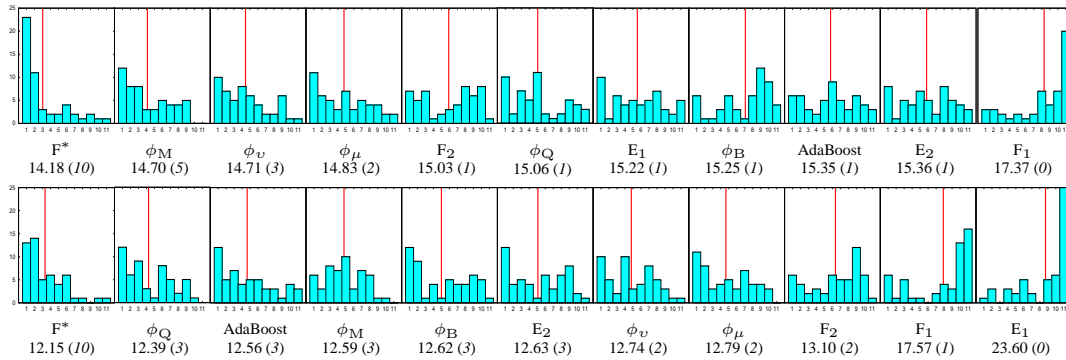

Figure 3: Summary of our results over the 52 domains for the 11 algorithms (top: $l = 2, r = 10$; bottom: $l = 3, r = 100$). Vertical (red) bars show the average rank over all domains (see text).

Three main conclusions emerge from Figure 3. First, $F^*$ appears to be superior to all other approaches, but slightly more sophisticated choices for the SCS (*i.e.* $E_., F_.$) fail at improving the results; this is a strong advocacy for a particular treatment of this *surrogate tuning problem*. Second, Matsushita's BCL, built from $\phi_M$, appears to be a serious alternative to the logistic loss. Third and last, a remark previously made by [10] for decision trees seems to hold as well for linear separators, as stronger concave regimes for $\phi$ in BCLs tend to improve performances at least for small $r$.

## Conclusion

In this paper, we have shown the existence of a supervised learning algorithm which minimizes any strictly convex, differentiable classification calibrated surrogate [3], inducing linear separators. Since the surrogate is now in the *input* of the algorithm, along with the learning sample, it opens the interesting problem of the tuning of this surrogate to the data at hand to further reduce the true risk. While the strategies we have experimentally tested are, with this respect, a simple primer for eventual solutions, they probably display the potential and the non triviality of these solutions.

## References

[1] A. Banerjee, X. Guo, and H. Wang. On the optimality of conditional expectation as a bregman predictor. *IEEE Trans. on Information Theory*, 51:2664–2669, 2005.

[2] A. Banerjee, S. Merugu, I. Dhillon, and J. Ghosh. Clustering with Bregman divergences. *Journal of Machine Learning Research*, 6:1705–1749, 2005.

[3] P. Bartlett, M. Jordan, and J. D. McAuliffe. Convexity, classification, and risk bounds. *Journal of the Am. Stat. Assoc.*, 101:138–156, 2006.

[4] P. Bartlett and M. Traskin. Adaboost is consistent. In *NIPS*19*, 2006.

[5] L. M. Bregman. The relaxation method of finding the common point of convex sets and its application to the solution of problems in convex programming. *USSR Comp. Math. and Math. Phys.*, 7:200–217, 1967.

[6] M. Collins, R. Schapire, and Y. Singer. Logistic regression, adaboost and Bregman distances. In *COLT'00*, pages 158–169, 2000.

[7] J. Friedman, T. Hastie, and R. Tibshirani. Additive Logistic Regression : a Statistical View of Boosting. *Ann. of Stat.*, 28:337–374, 2000.

[8] C. Gentile and M. Warmuth. Linear hinge loss and average margin. In *NIPS*11*, pages 225–231, 1998.

[9] P. Grünwald and P. Dawid. Game theory, maximum entropy, minimum discrepancy and robust Bayesian decision theory. *Ann. of Statistics*, 32:1367–1433, 2004.

[10] M.J. Kearns and Y. Mansour. On the boosting ability of top-down decision tree learning algorithms. *Journal of Comp. Syst. Sci.*, 58:109–128, 1999.

[11] K. Matsushita. Decision rule, based on distance, for the classification problem. *Ann. of the Inst. for Stat. Math.*, 8:67–77, 1956.

[12] R. Nock and F. Nielsen. A $\mathbb{R}$eal Generalization of discrete AdaBoost. *Artif. Intell.*, 171:25–41, 2007.

[13] R. E. Schapire and Y. Singer. Improved boosting algorithms using confidence-rated predictions. In *COLT'98*, pages 80–91, 1998.

[14] M. Warmuth, J. Liao, and G. Rätsch. Totally corrective boosting algorithms that maximize the margin. In *ICML'06*, pages 1001–1008, 2006.
